# Supervised learning from incomplete data via an EM approach

**Zoubin Ghahramani** and **Michael I. Jordan**
Department of Brain & Cognitive Sciences
Massachusetts Institute of Technology
Cambridge, MA 02139

## Abstract

Real-world learning tasks may involve high-dimensional data sets with arbitrary patterns of missing data. In this paper we present a framework based on maximum likelihood density estimation for learning from such data sets. We use mixture models for the density estimates and make two distinct appeals to the Expectation-Maximization (EM) principle (Dempster et al., 1977) in deriving a learning algorithm—EM is used both for the estimation of mixture components and for coping with missing data. The resulting algorithm is applicable to a wide range of supervised as well as unsupervised learning problems. Results from a classification benchmark—the iris data set—are presented.

## 1 Introduction

Adaptive systems generally operate in environments that are fraught with imperfections; nonetheless they must cope with these imperfections and learn to extract as much relevant information as needed for their particular goals. One form of imperfection is incompleteness in sensing information. Incompleteness can arise extrinsically from the data generation process and intrinsically from failures of the system's sensors. For example, an object recognition system must be able to learn to classify images with occlusions, and a robotic controller must be able to integrate multiple sensors even when only a fraction may operate at any given time.

In this paper we present a framework—derived from parametric statistics—for learn-

ing from data sets with arbitrary patterns of incompleteness. Learning in this framework is a classical estimation problem requiring an explicit probabilistic model and an algorithm for estimating the parameters of the model. A possible disadvantage of parametric methods is their lack of flexibility when compared with nonparametric methods. This problem, however, can be largely circumvented by the use of mixture models (McLachlan and Basford, 1988). Mixture models combine much of the flexibility of nonparametric methods with certain of the analytic advantages of parametric methods.

Mixture models have been utilized recently for supervised learning problems in the form of the "mixtures of experts" architecture (Jacobs et al., 1991; Jordan and Jacobs, 1994). This architecture is a parametric regression model with a modular structure similar to the nonparametric decision tree and adaptive spline models (Breiman et al., 1984; Friedman, 1991). The approach presented here differs from these regression-based approaches in that the goal of learning is to estimate the *density* of the data. No distinction is made between input and output variables; the joint density is estimated and this estimate is then used to form an input/output map. Similar approaches have been discussed by Specht (1991) and Tresp et al. (1993). To estimate the vector function $\mathbf{y} = f(\mathbf{x})$ the joint density $P(\mathbf{x}, \mathbf{y})$ is estimated and, given a particular input $\mathbf{x}$, the conditional density $P(\mathbf{y}|\mathbf{x})$ is formed. To obtain a single estimate of $\mathbf{y}$ rather than the full conditional density one can evaluate $\hat{\mathbf{y}} = E(\mathbf{y}|\mathbf{x})$, the expectation of $\mathbf{y}$ given $\mathbf{x}$.

The density-based approach to learning can be exploited in several ways. First, having an estimate of the joint density allows for the representation of any relation between the variables. From $P(\mathbf{x}, \mathbf{y})$, we can estimate $\hat{\mathbf{y}} = f(\mathbf{x})$, the inverse $\hat{\mathbf{x}} = f^{-1}(\mathbf{y})$, or any other relation between two subsets of the elements of the concatenated vector $(\mathbf{x}, \mathbf{y})$.

Second, this density-based approach is applicable both to supervised learning and unsupervised learning in exactly the same way. The only distinction between supervised and unsupervised learning in this framework is whether some portion of the data vector is denoted as "input" and another portion as "target".

Third, as we discuss in this paper, the density-based approach deals naturally with incomplete data, i.e. missing values in the data set. This is because the problem of estimating mixture densities can itself be viewed as a missing data problem (the "labels" for the component densities are missing) and an Expectation–Maximization (EM) algorithm (Dempster et al., 1977) can be developed to handle both kinds of missing data.

## 2    Density estimation using EM

This section outlines the basic learning algorithm for finding the maximum likelihood parameters of a mixture model (Dempster et al., 1977; Duda and Hart, 1973; Nowlan, 1991). We assume that the data $\mathcal{X} = \{\mathbf{x}_1, \ldots, \mathbf{x}_N\}$ are generated independently from a mixture density

$$P(\mathbf{x}_i) = \sum_{j=1}^{M} P(\mathbf{x}_i|\omega_j; \theta_j) P(\omega_j), \tag{1}$$

where each component of the mixture is denoted $\omega_j$ and parametrized by $\theta_j$. From equation (1) and the independence assumption we see that the log likelihood of the parameters given the data set is

$$l(\theta|\mathcal{X}) = \sum_{i=1}^{N} \log \sum_{j=1}^{M} P(\mathbf{x}_i|\omega_j;\theta_j)P(\omega_j). \tag{2}$$

By the maximum likelihood principle the best model of the data has parameters that maximize $l(\theta|\mathcal{X})$. This function, however, is not easily maximized numerically because it involves the log of a sum.

Intuitively, there is a "credit-assignment" problem: it is not clear which component of the mixture generated a given data point and thus which parameters to adjust to fit that data point. The EM algorithm for mixture models is an iterative method for solving this credit-assignment problem. The intuition is that if one had access to a "hidden" random variable $\mathbf{z}$ that indicated which data point was generated by which component, then the maximization problem would decouple into a set of simple maximizations. Using the indicator variable $\mathbf{z}$, a "complete-data" log likelihood function can be written

$$l_c(\theta|\mathcal{X},\mathcal{Z}) = \sum_{i=1}^{N}\sum_{j=1}^{M} z_{ij} \log P(\mathbf{x}_i|\mathbf{z}_i;\theta)P(\mathbf{z}_i;\theta), \tag{3}$$

which does not involve a log of a summation.

Since $\mathbf{z}$ is unknown $l_c$ cannot be utilized directly, so we instead work with its expectation, denoted by $Q(\theta|\theta_k)$. As shown by (Dempster et al., 1977), $l(\theta|\mathcal{X})$ can be maximized by iterating the following two steps:

$$
\begin{aligned}
E \ step{:} \quad Q(\theta|\theta_k) &= E[l_c(\theta|\mathcal{X},\mathcal{Z})|\mathcal{X},\theta_k] \\
M \ step{:} \quad \theta_{k+1} &= \underset{\theta}{\arg\max}\ Q(\theta|\theta_k).
\end{aligned} \tag{4}
$$

The E (Expectation) step computes the expected complete data log likelihood and the M (Maximization) step finds the parameters that maximize this likelihood. These two steps form the basis of the EM algorithm; in the next two sections we will outline how they can be used for real and discrete density estimation.

## 2.1   Real-valued data: mixture of Gaussians

Real-valued data can be modeled as a mixture of Gaussians. For this model the E-step simplifies to computing $h_{ij} \equiv E[z_{ij}|\mathbf{x}_i,\theta_k]$, the probability that Gaussian $j$, as defined by the parameters estimated at time step $k$, generated data point $i$.

$$h_{ij} = \frac{|\hat{\Sigma}_j^k|^{-1/2}\exp\{-\frac{1}{2}(\mathbf{x}_i - \hat{\mu}_j^k)^T \hat{\Sigma}_j^{-1,k}(\mathbf{x}_i - \hat{\mu}_j^k)\}}{\sum_{l=1}^{M}|\hat{\Sigma}_l^k|^{-1/2}\exp\{-\frac{1}{2}(\mathbf{x}_i - \hat{\mu}_l^k)^T \hat{\Sigma}_l^{-1,k}(\mathbf{x}_i - \hat{\mu}_l^k)\}}. \tag{5}$$

The M-step re-estimates the means and covariances of the Gaussians[1] using the data set weighted by the $h_{ij}$:

$$\text{a) } \hat{\mu}_j^{k+1} = \frac{\sum_{i=1}^{N} h_{ij}\mathbf{x}_i}{\sum_{i=1}^{N} h_{ij}}, \qquad \text{b) } \hat{\Sigma}_j^{k+1} = \frac{\sum_{i=1}^{N} h_{ij}(\mathbf{x}_i - \hat{\mu}_j^{k+1})(\mathbf{x}_i - \hat{\mu}_j^{k+1})^T}{\sum_{i=1}^{N} h_{ij}}. \tag{6}$$

## 2.2 Discrete-valued data: mixture of Bernoullis

D-dimensional binary data $\mathbf{x} = (x_1, \ldots, x_d, \ldots x_D)$, $x_d \in \{0, 1\}$, can be modeled as a mixture of $M$ Bernoulli densities. That is,

$$P(\mathbf{x}|\theta) = \sum_{j=1}^{M} P(\omega_j) \prod_{d=1}^{D} \mu_{jd}^{x_d} (1 - \mu_{jd})^{(1-x_d)}. \tag{7}$$

For this model the E-step involves computing

$$h_{ij} = \frac{\prod_{d=1}^{D} \hat{\mu}_{jd}^{x_{id}} (1 - \hat{\mu}_{jd})^{(1-x_{id})}}{\sum_{l=1}^{M} \prod_{d=1}^{D} \hat{\mu}_{ld}^{x_{id}} (1 - \hat{\mu}_{ld})^{(1-x_{id})}}, \tag{8}$$

and the M-step again re-estimates the parameters by

$$\hat{\mu}_{j}^{k+1} = \frac{\sum_{i=1}^{N} h_{ij} \mathbf{x}_i}{\sum_{i=1}^{N} h_{ij}}. \tag{9}$$

More generally, discrete or categorical data can be modeled as generated by a mixture of multinomial densities and similar derivations for the learning algorithm can be applied. Finally, the extension to data with mixed real, binary, and categorical dimensions can be readily derived by assuming a joint density with mixed components of the three types.

## 3 Learning from incomplete data

In the previous section we presented one aspect of the EM algorithm: learning mixture models. Another important application of EM is to learning from data sets with missing values (Little and Rubin, 1987; Dempster et al., 1977). This application has been pursued in the statistics literature for non-mixture density estimation problems; in this paper we combine this application of EM with that of learning mixture parameters.

We assume that the data set $\mathcal{X} = \{\mathbf{x}_1, \ldots, \mathbf{x}_N\}$ is divided into an observed component $\mathcal{X}^o$ and a missing component $\mathcal{X}^m$. Similarly, each data vector $\mathbf{x}_i$ is divided into $(\mathbf{x}_i^o, \mathbf{x}_i^m)$ where each data vector can have different missing components—this would be denoted by superscript $\mathbf{m}_i$ and $\mathbf{o}_i$, but we have simplified the notation for the sake of clarity.

To handle missing data we rewrite the EM algorithm as follows

$$
\begin{aligned}
\text{E step:} \quad & Q(\theta|\theta_k) = E[l_c(\theta|\mathcal{X}^o, \mathcal{X}^m, \mathcal{Z})|\mathcal{X}^o, \theta_k] \\
\text{M step:} \quad & \theta_{k+1} = \underset{\theta}{\arg\max} \; Q(\theta|\theta_k).
\end{aligned} \tag{10}
$$

Comparing to equation (4) we see that aside from the indicator variables $\mathcal{Z}$ we have added a second form of incomplete data, $\mathcal{X}^m$, corresponding to the missing values in the data set. The E-step of the algorithm estimates both these forms of missing information; in essence it uses the current estimate of the data density to complete the missing values.

## 3.1 Real-valued data: mixture of Gaussians

We start by writing the log likelihood of the complete data,

$$l_c(\theta|\mathcal{X}^o, \mathcal{X}^m, \mathcal{Z}) = \sum_i^N \sum_j^M z_{ij} \log P(\mathbf{x}_i|\mathbf{z}_i, \theta) + \sum_i^N \sum_j^M z_{ij} \log P(\mathbf{z}_i|\theta). \qquad (11)$$

We can ignore the second term since we will only be estimating the parameters of the $P(\mathbf{x}_i|\mathbf{z}_i, \theta)$. Using equation (11) for the mixture of Gaussians we note that if only the indicator variables $\mathbf{z}_i$ are missing, the E step can be reduced to estimating $E[z_{ij}|\mathbf{x}_i, \theta]$. For the case we are interested in, with two types of missing data $\mathbf{z}_i$ and $\mathbf{x}_i^m$, we expand equation (11) using m and o superscripts to denote subvectors and submatrices of the parameters matching the missing and observed components of the data,

$$l_c(\theta|\mathcal{X}^o, \mathcal{X}^m, \mathcal{Z}) = \sum_i^N \sum_j^M z_{ij} [\frac{n}{2} \log 2\pi + \frac{1}{2} \log |\Sigma_j| - \frac{1}{2}(\mathbf{x}_i^o - \mu_j^o)^T \Sigma_j^{-1,oo}(\mathbf{x}_i^o - \mu_j^o)$$

$$-(\mathbf{x}_i^o - \mu_j^o)^T \Sigma_j^{-1,om}(\mathbf{x}_i^m - \mu_j^m) - \frac{1}{2}(\mathbf{x}_i^m - \mu_j^m)^T \Sigma_j^{-1,mm}(\mathbf{x}_i^m - \mu_j^m)].$$

Note that after taking the expectation, the sufficient statistics for the parameters involve three unknown terms, $z_{ij}$, $z_{ij}\mathbf{x}_i^m$, and $z_{ij}\mathbf{x}_i^m\mathbf{x}_i^{mT}$. Thus we must compute: $E[z_{ij}|\mathbf{x}_i^o, \theta_k]$, $E[z_{ij}\mathbf{x}_i^m|\mathbf{x}_i^o, \theta_k]$, and $E[z_{ij}\mathbf{x}_i^m\mathbf{x}_i^{mT}|\mathbf{x}_i^o, \theta_k]$.

One intuitive approach to dealing with missing data is to use the current estimate of the data density to compute the expectation of the missing data in an E-step, complete the data with these expectations, and then use this completed data to re-estimate parameters in an M-step. However, this intuition fails even when dealing with a single two-dimensional Gaussian; the expectation of the missing data always lies along a line, which biases the estimate of the covariance. On the other hand, the approach arising from application of the EM algorithm specifies that one should use the current density estimate to compute the expectation of whatever incomplete terms appear in the likelihood maximization. For the mixture of Gaussians these incomplete terms involve interactions between the indicator variable $z_{ij}$ and the first and second moments of $\mathbf{x}_i^m$. Thus, simply computing the expectation of the missing data $\mathbf{z}_i$ and $\mathbf{x}_i^m$ from our model and substituting those values into the M step is *not* sufficient to guarantee an increase in the likelihood of the parameters.

The above terms can be computed as follows: $E[z_{ij}|\mathbf{x}_i^o, \theta_k]$ is again $h_{ij}$, the probability as defined in (5) measured only on the observed dimensions of $\mathbf{x}_i$, and

$$E[z_{ij}\mathbf{x}_i^m|\mathbf{x}_i^o, \theta_k] = h_{ij}E[\mathbf{x}_i^m|z_{ij} = 1, \mathbf{x}_i^o, \theta_k] = h_{ij}(\mu_j^m + \Sigma_j^{mo}\Sigma_j^{oo^{-1}}(\mathbf{x}_i^o - \mu_j^o)), \qquad (12)$$

Defining $\hat{\mathbf{x}}_{ij}^m \equiv E[\mathbf{x}_i^m|z_{ij} = 1, \mathbf{x}_i^o, \theta_k]$, the regression of $\mathbf{x}_i^m$ on $\mathbf{x}_i^o$ using Gaussian $j$,

$$E[z_{ij}\mathbf{x}_i^m\mathbf{x}_i^{mT}|\mathbf{x}_i^o, \theta_k] = h_{ij}(\Sigma_j^{mm} - \Sigma_j^{mo}\Sigma_j^{oo^{-1}}\Sigma_j^{moT} + \hat{\mathbf{x}}_{ij}^m\hat{\mathbf{x}}_{ij}^{mT}). \qquad (13)$$

The M-step uses these expectations substituted into equations (6)a and (6)b to re-estimate the means and covariances. To re-estimate the mean vector, $\mu_j$, we substitute the values $E[\mathbf{x}_i^m|z_{ij} = 1, \mathbf{x}_i^o, \theta_k]$ for the missing components of $\mathbf{x}_i$ in equation (6)a. To re-estimate the covariance matrix we substitute the values $E[\mathbf{x}_i^m\mathbf{x}_i^{mT}|z_{ij} = 1, \mathbf{x}_i^o, \theta_k]$ for the outer product matrices involving the missing components of $\mathbf{x}_i$ in equation (6)b.

## 3.2 Discrete-valued data: mixture of Bernoullis

For the Bernoulli mixture the sufficient statistics for the M-step involve the incomplete terms $E[z_{ij}|\mathbf{x}_i^o, \theta_k]$ and $E[z_{ij}\mathbf{x}_i^m|\mathbf{x}_i^o, \theta_k]$. The first is equal to $h_{ij}$ calculated over the observed subvector of $\mathbf{x}_i$. The second, since we assume that within a class the individual dimensions of the Bernoulli variable are independent, is simply $h_{ij}\boldsymbol{\mu}_j^m$. The M-step uses these expectations substituted into equation (9).

# 4 Supervised learning

If each vector $\mathbf{x}_i$ in the data set is composed of an "input" subvector, $\mathbf{x}_i^i$, and a "target" or output subvector, $\mathbf{x}_i^o$, then learning the joint density of the input and target is a form of supervised learning. In supervised learning we generally wish to predict the output variables from the input variables. In this section we will outline how this is achieved using the estimated density.

## 4.1 Function approximation

For real-valued function approximation we have assumed that the density is estimated using a mixture of Gaussians. Given an input vector $\mathbf{x}_i^i$ we extract all the relevant information from the density $P(\mathbf{x}^i, \mathbf{x}^o)$ by conditionalizing to $P(\mathbf{x}^o|\mathbf{x}_i^i)$. For a single Gaussian this conditional density is normal, and, since $P(\mathbf{x}^i, \mathbf{x}^o)$ is a mixture of Gaussians so is $P(\mathbf{x}^o|\mathbf{x}^i)$. In principle, this conditional density is the final output of the density estimator. That is, given a particular input the network returns the complete conditional density of the output. However, since many applications require a single estimate of the output, we note three ways to obtain estimates $\hat{\mathbf{x}}$ of $\mathbf{x}^o = f(\mathbf{x}_i^i)$: the least squares estimate (LSE), which takes $\hat{\mathbf{x}}^o(\mathbf{x}_i^i) = E(\mathbf{x}^o|\mathbf{x}_i^i)$; stochastic sampling (STOCH), which samples according to the distribution $\hat{\mathbf{x}}^o(\mathbf{x}_i^i) \sim P(\mathbf{x}^o|\mathbf{x}_i^i)$; single component LSE (SLSE), which takes $\hat{\mathbf{x}}^o(\mathbf{x}_i^i) = E(\mathbf{x}^o|\mathbf{x}_i^i, \omega_j)$ where $j = \arg\max_k P(z_k|\mathbf{x}_i^i)$. For a given input, SLSE picks the Gaussian with highest posterior and approximates the output with the LSE estimator given by that Gaussian alone.

The conditional expectation or LSE estimator for a Gaussian mixture is

$$\hat{\mathbf{x}}^o(\mathbf{x}_i^i) = \frac{\sum_{j=1}^{M} h_{ij}[\boldsymbol{\mu}_j^o + \Sigma_j^{oi}\Sigma_j^{ii^{-1}}(\mathbf{x}_i^i - \boldsymbol{\mu}_j^i)]}{\sum_{j=1}^{M} h_{ij}}, \qquad (14)$$

which is a convex sum of linear approximations, where the weights $h_{ij}$ vary nonlinearly according to equation (14) over the input space. The LSE estimator on a Gaussian mixture has interesting relations to algorithms such as CART (Breiman et al., 1984), MARS (Friedman, 1991), and mixtures of experts (Jacobs et al., 1991; Jordan and Jacobs, 1994), in that the mixture of Gaussians competitively partitions the input space, and learns a linear regression surface on each partition. This similarity has also been noted by Tresp et al. (1993) .

The stochastic estimator (STOCH) and the single component estimator (SLSE) are better suited than any least squares method for learning non-convex inverse maps, where the mean of several solutions to an inverse might not be a solution. These

**Classification with missing inputs**

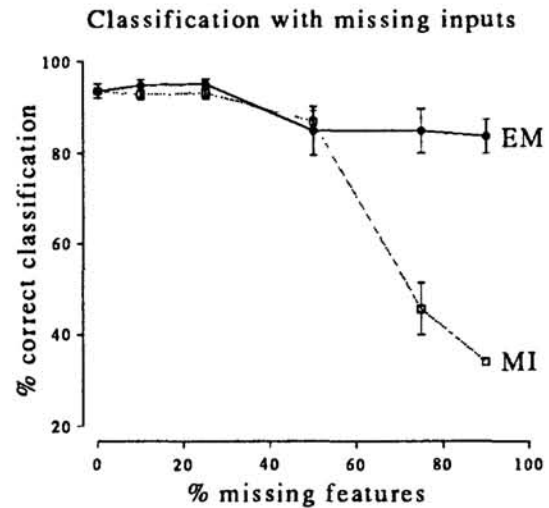

Figure 1: Classification of the iris data set. 100 data points were used for training and 50 for testing. Each data point consisted of 4 real-valued attributes and one of three class labels. The figure shows classification performance $\pm$ 1 standard error ($n = 5$) as a function of proportion missing features for the EM algorithm and for mean imputation (MI), a common heuristic where the missing values are replaced with their unconditional means.

estimators take advantage of the explicit representation of the input/output density by selecting one of the several solutions to the inverse.

## 4.2   Classification

Classification problems involve learning a mapping from an input space into a set of discrete class labels. The density estimation framework presented in this paper lends itself to solving classification problems by estimating the joint density of the input and class label using a mixture model. For example, if the inputs have real-valued attributes and there are $D$ class labels, a mixture model with Gaussian and multinomial components will be used:

$$P(\mathbf{x}, \mathcal{C} = d|\theta) = \sum_{j=1}^{M} P(\omega_j) \frac{\mu_{jd}}{(2\pi)^{n/2}|\Sigma_j|^{1/2}} \exp\{-\frac{1}{2}(\mathbf{x} - \mu_j)^T \Sigma_j^{-1}(\mathbf{x} - \mu_j)\}, \quad (15)$$

denoting the joint probability that the data point is $\mathbf{x}$ and belongs to class $d$, where the $\mu_{jd}$ are the parameters for the multinomial. Once this density has been estimated, the maximum likelihood label for a particular input $\mathbf{x}$ may be obtained by computing $P(\mathcal{C} = d|\mathbf{x}, \theta)$. Similarly, the class conditional densities can be derived by evaluating $P(\mathbf{x}|\mathcal{C} = d, \theta)$. Conditionalizing over classes in this way yields class conditional densities which are in turn mixtures of Gaussians. Figure 1 shows the performance of the EM algorithm on an example classification problem with varying proportions of missing features. We have also applied these algorithms to the problems of clustering 35-dimensional greyscale images and approximating the kinematics of a three-joint planar arm from incomplete data.

## 5   Discussion

Density estimation in high dimensions is generally considered to be more difficult—requiring more parameters—than function approximation. The density-estimation-based approach to learning, however, has two advantages. First, it permits ready incorporation of results from the statistical literature on missing data to yield flexible supervised and unsupervised learning architectures. This is achieved by combining two branches of application of the EM algorithm yielding a set of learning rules for mixtures under incomplete sampling.

Second, estimating the density explicitly enables us to represent any relation between the variables. Density estimation is fundamentally more general than function approximation and this generality is needed for a large class of learning problems arising from inverting causal systems (Ghahramani, 1994). These problems cannot be solved easily by traditional function approximation techniques since the data is not generated from noisy samples of a function, but rather of a relation.

## Acknowledgements

Thanks to D. M. Titterington and David Cohn for helpful comments. This project was supported in part by grants from the McDonnell-Pew Foundation, ATR Auditory and Visual Perception Research Laboratories, Siemens Corporation, the National Science Foundation, and the Office of Naval Research. The iris data set was obtained from the UCI Repository of Machine Learning Databases.

## Footnotes

[1]Though this derivation assumes equal priors for the Gaussians, if the priors are viewed as mixing parameters they can also be learned in the maximization step.

# References

Breiman, L., Friedman, J. H., Olshen, R. A., and Stone, C. J. (1984). *Classification and Regression Trees.* Wadsworth International Group, Belmont, CA.

Dempster, A. P., Laird, N. M., and Rubin, D. B. (1977). Maximum likelihood from incomplete data via the EM algorithm. *J. Royal Statistical Society Series B,* 39:1–38.

Duda, R. O. and Hart, P. E. (1973). *Pattern Classification and Scene Analysis.* Wiley, New York.

Friedman, J. H. (1991). Multivariate adaptive regression splines. *The Annals of Statistics,* 19:1–141.

Ghahramani, Z. (1994). Solving inverse problems using an EM approach to density estimation. In *Proceedings of the 1993 Connectionist Models Summer School.* Erlbaum, Hillsdale, NJ.

Jacobs, R., Jordan, M., Nowlan, S., and Hinton, G. (1991). Adaptive mixture of local experts. *Neural Computation,* 3:79–87.

Jordan, M. and Jacobs, R. (1994). Hierarchical mixtures of experts and the EM algorithm. *Neural Computation,* 6:181–214.

Little, R. J. A. and Rubin, D. B. (1987). *Statistical Analysis with Missing Data.* Wiley, New York.

McLachlan, G. and Basford, K. (1988). *Mixture models: Inference and applications to clustering.* Marcel Dekker.

Nowlan, S. J. (1991). *Soft Competitive Adaptation: Neural Network Learning Algorithms based on Fitting Statistical Mixtures.* CMU-CS-91-126, School of Computer Science, Carnegie Mellon University, Pittsburgh, PA.

Specht, D. F. (1991). A general regression neural network. *IEEE Trans. Neural Networks,* 2(6):568–576.

Tresp, V., Hollatz, J., and Ahmad, S. (1993). Network structuring and training using rule-based knowledge. In Hanson, S. J., Cowan, J. D., and Giles, C. L., editors, *Advances in Neural Information Processing Systems 5.* Morgan Kaufman Publishers, San Mateo, CA.
